# Neural Network Based Model Predictive Control

**Stephen Piché**
Pavilion Technologies
Austin, TX 78758
*spiche@pav.com*

**Jim Keeler**
Pavilion Technologies
Austin, TX 78758
*jkeeler@pav.com*

**Greg Martin**
Pavilion Technologies
Austin, TX 78758
*gmartin@pav.com*

**Gene Boe**
Pavilion Technologies
Austin, TX 78758
*gboe@pav.com*

**Doug Johnson**
Pavilion Technologies
Austin, TX 78758
*djohnson@pav.com*

**Mark Gerules**
Pavilion Technologies
Austin, TX 78758
*mgerules@pav.com*

## Abstract

Model Predictive Control (MPC), a control algorithm which uses an optimizer to solve for the optimal control moves over a future time horizon based upon a model of the process, has become a standard control technique in the process industries over the past two decades. In most industrial applications, a linear dynamic model developed using empirical data is used even though the process itself is often nonlinear. Linear models have been used because of the difficulty in developing a generic nonlinear model from empirical data and the computational expense often involved in using nonlinear models. In this paper, we present a generic neural network based technique for developing nonlinear dynamic models from empirical data and show that these models can be efficiently used in a model predictive control framework. This nonlinear MPC based approach has been successfully implemented in a number of industrial applications in the refining, petrochemical, paper and food industries. Performance of the controller on a nonlinear industrial process, a polyethylene reactor, is presented.

## 1 Introduction

Model predictive control has become the standard technique for supervisory control in the process industries with over 2,000 applications in the refining, petrochemicals, chemicals, pulp and paper, and food processing industries [1]. Model Predictive Control was developed in the late 70's and came into wide-spread use, particularly in the refining industry, in the 80's. The economic benefit of this approach to control has been documented [1,2].

Several factors have contributed to the wide-spread use of MPC in the process industries:

1. *Multivariate Control*: Industrial processes are typically coupled multiple-input multiple-output (MIMO) systems. MIMO control can be implemented using MPC.

2. *Constraints*: Constraints on the inputs and outputs of a process due to safety considerations are common in the process industries. These constraints can be integrated into the control calculation using MPC.

3. *Sampling Period*: Unlike systems in other industries such as automotive or aerospace, the open-loop settling times for many processes is on the order of hours rather than milliseconds. This slow settling time translates to sampling periods on the order of minutes. Because the sampling period is sufficiently long, the complex optimization calculations that are required to implement MPC can be solved at each sampling period.

4. *Commercial Tools:* Commercial tools that facilitate model development and controller implementation have allowed proliferation of MPC in the process industries.

Until recently, industrial applications of MPC have relied upon linear dynamic models even though most processes are nonlinear. MPC based upon linear models is acceptable when the process operates at a single setpoint and the primary use of the controller is the rejection of disturbances. However, many chemical processes, including polymer reactors, do not operate at a single setpoint. These processes are often required to operate at different setpoints depending upon the grade of the product that is to be produced. Because these processes operate over the nonlinear range of the system, linear MPC often results in poor performance. To properly control these processes, a nonlinear model is needed in the MPC algorithm.

This need for nonlinear models in MPC is well recognized. A number of researchers and commercial companies have developed both simulation and industrial applications using a variety of different technologies including both first principles and empirical approaches such as neural networks [3,4]. Although a variety of different models have been developed, they have not been practical for wide scale industrial application. On one hand, nonlinear models built using first principle techniques are expensive to develop and are specific to a process. Conversely, many empirically based nonlinear models are not appropriate for wide scale use because they require costly plant tests in multiple operating regions or because they are too computationally expensive to use in a real-time environment.

This paper presents a nonlinear model that has been developed for wide scale industrial use. It is an empirical model based upon a neural network which is developed using plant test data from a single operating region and historical data from all regions. This is in contrast to the usual approach of using plant test data from multiple regions. This model has been used on over 50 industrial applications and was recognized in a recent survey paper on nonlinear MPC as the most widely used nonlinear MPC controller in the process industries[1].

After providing a brief overview of model predictive control in the next section, we present details on the formulation of the nonlinear model. After describing the model, an industrial application is presented that validates the usefulness of the nonlinear model in an MPC algorithm.

## 2   Model Predictive Control

Model predictive control is based upon solving an optimization problem for the control actions at each sampling interval. Using MPC, an optimizer computes future control actions that minimize the difference between a model of the process and desired performance over a time horizon (typically the time horizon is greater than the open-loop settling time of the process). For example, given a linear model of process,

$$y_t = -a_1 y_{t-1} - a_2 y_{t-2} + b_1 u_{t-1} + b_2 u_{t-2} \tag{1}$$

where $u(t)$ represents the input to the process, the optimizer may be used to minimize an objective function at time $t$,

$$J = \sum_{i=1}^{T} ((y_{t+i} - \hat{y}_{t+i})^2 + (u_{t+i} - u_{t+i-1})^2) \tag{2}$$

where $\hat{y}_t$ is the desired setpoint for the output and $T$ is the length of the time horizon. In addition to minimizing an objective function, the optimizer is used to observe a set of constraints. For example, it is common to place upper and lower bounds on the inputs as well as bounds on the rate of change of the input,

$$U_{upper} \geq u_{t+i} \geq U_{lower} \quad \forall \ \ 1 \leq i \leq T \tag{3}$$

$$\Delta U_{upper} \geq u_{t+i} - u_{t+i-1} \geq \Delta U_{lower} \quad \forall \ \ 1 \leq i \leq T \tag{4}$$

where $U_{upper}$ and $U_{lower}$ are the upper and lower input bounds while $\Delta U_{upper}$ and $\Delta U_{lower}$ are the upper and lower rate of change bounds. After the trajectory of future control actions is computed, only the first value in the trajectory is sent as a setpoint to the actuators. The optimization calculation is re-run at each sampling interval using a model which has been updated using feedback.

The form of the model, the objective function, the constraints and the type of optimizer have been active areas of research over the past two decades. A number of excellent survey papers on MPC cover these topics [1,2,4]. As discussed above, we have selected a MIMO nonlinear model which is presented in the next section. Although the objective function given above contains two terms (desired output and input move suppression), the objective function used in our implementation contains thirteen separate terms. (The details of the objective function are beyond the scope of this paper.) Our implementation uses the constraints given above in (3) and (4). Because we use nonlinear models, a nonlinear programming technique must be used to solve the optimization problem. We use LS-GRG which is a reduced gradient solver [5].

# 3   A Generic and Parsimonious Nonlinear Model

For a nonlinear model to achieve wide-spread industrial use, the model must be parsimonious so that it can be efficiently used in an optimization problem. Furthermore, it must be developed from limited process data. As discussed below, the nonlinear model we use is composed of a combination of a nonlinear steady state model and a linear dynamic model which can be derived from available data. The method of combining the models results in a parsimonious nonlinear model.

## 3.1   Process data and component models

The quantity and quality of available data ultimately determines the structure of an empirical model. In developing our models, the available data dictated the type of model that could be created. In the process industries, two types of data are available:

1. *Historical data:* The values of the inputs and outputs of most processes are saved at regular intervals to a data base. Furthermore, most processing companies retain historical data associated with their plant for several years.

2. *Plant tests:* Open-loop testing is a well accepted practice for determining the process dynamics for implementation of MPC. However, open-loop testing in multiple operating regions is not well accepted and is impractical in most cases even if it were accepted.

Most practitioners of MPC models have used plant test data and ignored historical data. Practitioners have ignored the historical data in the past because it was difficult to extract and preprocess the data, and build models. Historical data was also viewed as not useful because it was collected in closed-loop and therefore process dynamics could not be extracted in many cases. Using only the plant test data, the practitioner is limited to linear dynamic models.

We chose to use the historical data because it can be used to create nonlinear steady state models of processes that operate at multiple setpoints. Combining the nonlinear steady state model with linear dynamic models from the plant test data provides a generic approach to developing nonlinear models.

To easily facilitate the development of nonlinear models, a suite of tools has been developed for data extraction and preprocessing as well as model training. The nonlinear steady state models,

$$\mathbf{y}_{ss} = NN_{ss}(\mathbf{u}) \tag{5}$$

are implemented by a feedforward neural network and trained using variants of the backpropagation algorithm [6]. The developer has a great deal of flexibility in determining the architecture of the network including the ability to select which inputs affect which outputs. Finally, an algorithm for specifying bounds on the gain (Jacobian) of the model has recently been implemented [7].

Because of limited plant test data, the dynamic models are restricted to second order models with input time delay,

$$y_t = -a_1 y_{t-1} - a_2 y_{t-2} + b_1 u_{t-d-1} + b_2 u_{t-d-2} \tag{6}$$

The parameters of (6) are identified by minimizing the squared error between the model and the plant test data. To prevent a biased estimate of the parameters, the identification problem is solved using an optimizer because of the correlation in the model inputs [8]. Tools for selecting the identification regions and viewing the results are provided.

## 3.2 Combining the nonlinear steady state and dynamic models

A variety of techniques exist for combining nonlinear steady state and linear dynamic models. The dynamic models can be used to either preprocess the inputs or postprocess the outputs of the steady state model. These models, referred to as Hammerstein and Weiner models respectively [8], contain a large number of parameters and are computationally expensive in an optimization problem when the model has many inputs and outputs. These models, when based upon neural networks, also extrapolate poorly.

Gain scheduling is often used to combine nonlinear steady state models and linear dynamic models. Using a neural network steady state model, the gain at the current operating point, $u_i$,

$$g_i = \frac{\partial y_{ss}}{\partial u}\mid_{u=u_i} \tag{7}$$

is used to update the gain of the linear dynamic model of (6),

$$\delta y_t = -a_1\delta y_{t-1} - a_2\delta y_{t-2} + v_1\delta u_{t-d-1} + v_2\delta u_{t-d-2} \tag{8}$$

where

$$v_1 = b_1 g_i \frac{1 + a_1 + a_2}{b_1 + b_2} \tag{9}$$

$$v_2 = b_2 g_i \frac{1 + a_1 + a_2}{b_1 + b_2}. \tag{10}$$

The difference equation is linearized about the point $u_i$ and $y_i = NN(u_i)$, thus, $\delta y = y - y_i$ and $\delta u = u - u_i$. To simplify the equations above, a single-input single-output (SISO) system is used. Gain scheduling results in a parsimonious model that is efficient to use in the MPC optimization problem, however, because this model does not incorporate information about the *gain over the entire trajectory*, its use leads to suboptimal performance in the MPC algorithm.

Our nonlinear model approach remedies this problem. By solving a steady state optimization problem whenever a setpoint change is made, it is possible to compute the final steady state values of the inputs, $u_f$. Given the final steady state input values, the gain associated with the final steady state can be computed. For a SISO system, this gain is given by

$$g_f = \frac{\partial y_{ss}}{\partial u}\mid_{u=u_f}. \tag{11}$$

Using the initial and final gain associated with a setpoint change, the gain structure over the entire trajectory can be approximated. This two point gain scheduling overcomes the limitations of regular gain scheduling in MPC algorithms.

Combining the initial and final gain with the linear dynamic model, a quadratic difference equation is derived for the overall nonlinear model,

$$\delta y_t = -a_1 \delta y_{t-1} - a_2 \delta y_{t-2} + v_1 \delta u_{t-d-1} + v_2 \delta u_{t-d-2} + w_1 \delta u_{t-d-1}^2 + w_2 \delta u_{t-d-2}^2 \quad (12)$$

where

$$w_1 = b_1 \frac{(1 + a_1 + a_2)(g_f - g_i)}{(b_1 + b_2)(u_f - u_i)} \quad (13)$$

$$w_2 = b_2 \frac{(1 + a_1 + a_2)(g_f - g_i)}{(b_1 + b_2)(u_f - u_i)} \quad (14)$$

and $v_1$ and $v_2$ are given by (9) and (10). Use of the gain at the final steady state introduces the last two terms of (12). This model allows the incorporation of gain information over the entire trajectory in the MPC algorithm. The gain at of (12) at $u_i$ is $g_i$ while at $u_f$ it is $g_f$. Between the two points, the gain is a linear combination of $g_i$ and $g_f$. For processes with large gain changes, such as polymer reactors, this can lead to dramatic improvements in MPC controller performance.

An additional benefit of using the model of (12) is that we allow the user to bound the initial and final gain and thus control the amount of nonlinearity used in the model. For practitioners who are use to implementing MPC with linear models, using gain bounds allows them to transition from linear to nonlinear models. This ability to control the amount of nonlinearity used in the model has been important for acceptance of this new model in many applications. Finally, bounding the gains can be used to guarantee extrapolation performance of the model.

The nonlinear model of (12) fits the criteria needed in order to allow wide spread use of nonlinear models for MPC. The model is based upon readily available data and has a parsimonious representation allowing models with many inputs and outputs to be efficiently used in the optimizer. Furthermore, it addresses the primary nonlinearity found in processes, that being the significant change in gain over the operating region.

## 4   Polymer Application

The nonlinear model described above has been used in a wide-variety of industrial applications including Kamyr digesters (pulp and paper), milk evaporators and dryers (food processing), toluene diamine purification (chemicals), polyethylene and polypropylene reactors (polymers) and a fluid catalytic cracking unit (refining). Highlights of one such application are given below.

A MPC controller that uses the model described above has been applied to a Gas Phase High Density Polyethylene reactor at Chevron Chemical Co. in Cedar Bayou, Texas [9]. The process produces homopolymer and copolymer grades over a wide range of melt indices. It's average production rate per year is 230,000 tons.

Optimal control of the process is difficult to achieve because the reactor is a highly coupled nonlinear MIMO system (7 inputs and 5 outputs). For example, a number of input-output pairs exhibit gains that varying by a factor of 10 or more over the operating region. In addition, grade changes are made every few days. During these transitions nonprime polymer is produced. Prior to commissioning these controllers,

these transitions took several hours to complete. Linear and gain scheduling based controller have been tried on similar reactors and have delivered limited success.

The nonlinear model was constructed using only historical data. The nonlinear steady state model was trained upon historical data from a two year period. This data contained examples of all the products produced by the reactor. Accurate dynamic models were derived both from historical data and knowledge of the process, thus, no step tests were conducted on the process.

Excellent performance of this controller has been reported [9]. A two-fold decrease in the variance of the primary quality variable (melt index) has been achieved. In addition, the average transition time has been decreased by 50%. Unscheduled shutdowns which occurred previously have been eliminated. Finally, the controller, which has been on-line for two years, has gained high operator acceptance.

## 5  Conclusion

A generic and parsimonious nonlinear model which can be used in an MPC algorithm has been presented. The model is created by combining a nonlinear steady state model with a linear dynamic models. They are combined using a two-point gain scheduling technique. This nonlinear model has been used for control of a nonlinear MIMO polyethylene reactor at Chevron Chemical Co. The controller has also been used in 50 other applications in the refining, chemicals, food processing and pulp and paper industries.

## References

[1] Qin, S.J. & Badgwell, T.A. (1997) An overview of industrial model predictive control technology. In J. Kantor, C. Garcia and B. Carnahan (eds.), *Chemical Process Control - AIChE Symposium Series*, pp. 232-256. NY: AIChE.

[2] Seborg, D.E. (1999) A perspective on advanced strategies for Process Control (Revisited). to appear in *Proc. of European Control Conf.* Karlsruhe, Germany.

[3] Qin, S.J. & Badgwell, T.A. (1998) An overview of nonlinear model predictive control applications. *Proc. IFAC Workshop on Nonlinear Model Predictive Control - Assessment and Future Directions*, Ascona, Switzerland, June 3-5.

[4] Meadow, E.S. & Rawlings, J.B. (1997) Model predictive control. In M. Hesnon and D. Seborg (eds.), *Nonlinear Model Predictive Control*, pp. 233-310. NJ: Prentice Hall.

[5] Nash, S. & Sofer, A. (1996) *Linear and Nonlinear Programming.* NY: McGraw-Hill.

[6] Rumelhart D.E, Hinton G.E. & Williams, R.J. (1986) Learning internal representations by error propagation. In D. Rumelhart and J. McClelland (eds.), *Parallel Distributed Processing*, pp. 318-362. Cambridge, MA: MIT Press.

[7] Hartman, E. (2000) Training feedforward neural networks with gain constraints. To appear in *Neural Computation.*

[8] Ljung, L. (1987) *System Identification.* NJ: Prentice Hall.

[9] Goff S., Johnson D. & Gerules, M. (1998) Nonlinear control and optimization of a high density polyethylene reactor. *Proc. Chemical Engineering Expo*, Houston, June.
